# Synchrony and Desynchrony in Neural Oscillator Networks

**DeLiang Wang**
Department of Computer and Information Science
and Center for Cognitive Science
The Ohio State University
Columbus, Ohio 43210, USA
dwang@cis.ohio-state.edu

**David Terman**
Department of Mathematics
The Ohio State University
Columbus, Ohio 43210, USA
terman@math.ohio-state.edu

## Abstract

An novel class of locally excitatory, globally inhibitory oscillator networks is proposed. The model of each oscillator corresponds to a standard relaxation oscillator with two time scales. The network exhibits a mechanism of selective gating, whereby an oscillator jumping up to its active phase rapidly recruits the oscillators stimulated by the same pattern, while preventing others from jumping up. We show analytically that with the selective gating mechanism the network rapidly achieves both synchronization within blocks of oscillators that are stimulated by connected regions and desynchronization between different blocks. Computer simulations demonstrate the network's promising ability for segmenting multiple input patterns in real time. This model lays a physical foundation for the oscillatory correlation theory of feature binding, and may provide an effective computational framework for scene segmentation and figure/ground segregation.

## 1 INTRODUCTION

A basic attribute of perception is its ability to group elements of a perceived scene into coherent clusters (objects). This ability underlies perceptual processes such as figure/ground segregation, identification of objects, and separation of different objects, and it is generally known as scene segmentation or perceptual organization. Despite the fact

that humans perform it with apparent ease, the general problem of scene segmentation remains unsolved in the engineering of sensory processing, such as computer vision and auditory processing.

Fundamental to scene segmentation is the grouping of similar sensory features and the segregation of dissimilar ones. Theoretical investigations of brain functions and feature binding point to the mechanism of temporal correlation as a representational framework (von der Malsburg, 1981; von der Malsburg and Schneider, 1986). In particular, the correlation theory of von der Malsburg (1981) asserts that an object is represented by the temporal correlation of the firing activities of the scattered cells coding different features of the object. A natural way of encoding temporal correlation is to use neural oscillations, whereby each oscillator encodes some feature (maybe just a pixel) of an object. In this scheme, each segment (object) is represented by a group of oscillators that shows synchrony (phase-locking) of the oscillations, and different objects are represented by different groups whose oscillations are desynchronized from each other. Let us refer to this form of temporal correlation as *oscillatory correlation*. The theory of oscillatory correlation has received direct experimental support from the cell recordings in the cat visual cortex (Eckhorn et al., 1988; Gray et al., 1989) and other brain regions. The discovery of synchronous oscillations in the visual cortex has triggered much interest from the theoretical community in simulating the experimental results and in exploring oscillatory correlation to solve the problems of scene segmentation. While several demonstrate synchronization in a group of oscillators using local (lateral) connections (König and Schillen, 1991; Somers and Kopell, 1993; Wang, 1993, 1995), most of these models rely on long range connections to achieve phase synchrony. It has been pointed out that local connections in reaching synchrony may play a fundamental role in scene segmentation since long-range connections would lead to indiscriminate segmentation (Sporns et al., 1991; Wang, 1993).

There are two aspects in the theory of oscillatory correlation: (1) synchronization within the same object; and (2) desynchronization between different objects. Despite intensive studies on the subject, the question of desynchronization has been hardly addressed. The lack of an efficient mechanism for desynchronization greatly limits the utility of oscillatory correlation to perceptual organization. In this paper, we propose a new class of oscillatory networks and show that it can rapidly achieve both synchronization within each object and desynchronization between a number of simultaneously presented objects. The network is composed of the following elements: (1) A new model of a basic oscillator; (2) Local excitatory connections to produce phase synchrony within each object; (3) A global inhibitor that receives inputs from the entire network and feeds back with inhibition to produce desynchronization of the oscillator groups representing different objects. In other words, the mechanism of the network consists of local cooperation and global competition. This surprisingly simple neural architecture may provide an elementary approach to scene segmentation and a computational framework for perceptual organization.

## 2  MODEL DESCRIPTION

The building block of this network, a single oscillator $i$, is defined in the simplest form as a feedback loop between an excitatory unit $x_i$ and an inhibitory unit $y_i$:

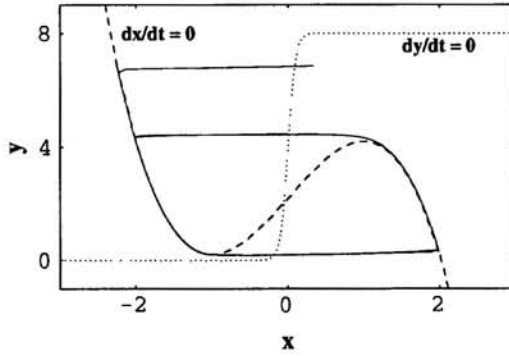
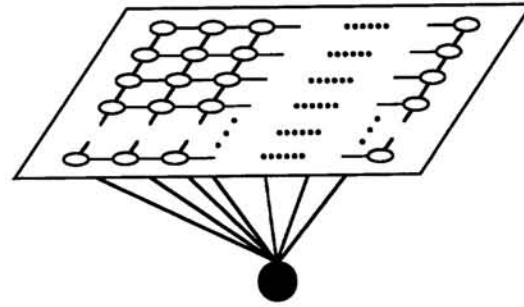

Figure 1: Nullclines and periodic orbit of a single oscillator as shown in the phase plane. When the oscillator starts at a randomly generated point in the phase plane, it quickly converged to a stable trajectory of a limit cycle.

Figure 2: Architecture of a two dimensional network with nearest neighbor coupling. The global inhibitor is indicated by the black circle.

$$\frac{dx_i}{dt} = 3x_i - x_i^3 + 2 - y_i + \rho + I_i + S_i \tag{1a}$$

$$\frac{dy_i}{dt} = \varepsilon \left( \gamma (1 + tanh(x_i/\beta)) - y_i \right) \tag{1b}$$

where $\rho$ denotes the amplitude of a Gaussian noise term. $I_i$ represents external stimulation to the oscillator, and $S_i$ denotes coupling from other oscillators in the network. The noise term is introduced both to test the robustness of the system and to actively desynchronize different input patterns. The parameter $\varepsilon$ is chosen to be small. In this case (1), without any coupling or noise, corresponds to a standard relaxation oscillator. The x-nullcline of (1) is a cubic curve, while the y-nullcline is a sigmoid function, as shown in Fig. 1. If $I > 0$, these curves intersect along the middle branch of the cubic, and (1) is oscillatory. The periodic solution alternates between the silent and active phases of near steady state behavior. The parameter $\gamma$ is introduced to control the relative times that the solution spends in these two phases. If $I < 0$, then the nullclines of (1) intersect at a stable fixed point along the left branch of the cubic. In this case the system produces no oscillation. The oscillator model (1) may be interpreted as a model of spiking behavior of a single neuron, or a mean field approximation to a network of excitatory and inhibitory neurons.

The network we study here in particular is two dimensional. However, the results can easily be extended to other dimensions. Each oscillator in the network is connected to only its four nearest neighbors, thus forming a 2-D grid. This is the simplest form of local connections. The global inhibitor receives excitation from each oscillator of the grid, and in turn inhibits each oscillator. This architecture is shown in Fig. 2. The intuitive reason why the network gives rise to scene segmentation is the following. When multiple connected objects are mapped onto the grid, local connectivity on the grid will group together the oscillators covered by each object. This grouping will be reflected

by phase synchrony within each object. The global inhibitor is introduced for desynchronizing the oscillatory responses to different objects. We assume that the coupling term $S_i$ in (1) is given by

$$S_i = \sum_{k \in N(i)} W_{ik}\, S_\infty(x_k,\, \theta_x) - W_z\, S_\infty(z,\, \theta_{xz}) \tag{2}$$

$$S_\infty(x,\, \theta) = \frac{1}{1 + exp[-K(x-\theta)]} \tag{3}$$

where $W_{ik}$ is a connection (synaptic) weight from oscillator $k$ to oscillator $i$, and $N(i)$ is the set of the neighoring oscillators that connect to $i$. In this model, $N(i)$ is the four immediate neighbors on the 2-D grid, except on the boundaries where $N(i)$ may be either 2 or 3 immediate neighbors. $\theta_x$ is a threshold (see the sigmoid function of Eq. 3) above which an oscillator can affect its neighbors. $W_z$ (positive) is the weight of inhibition from the global inhibitor $z$, whose activity is defined as

$$\frac{dz}{dt} = \phi\,(\sigma_\infty - z) \tag{4}$$

where $\sigma_\infty = 0$ if $x_i < \theta_{zx}$ for every oscillator, and $\sigma_\infty = 1$ if $x_i \geq \theta_{zx}$ for at least one oscillator $i$. Hence $\theta_{zx}$ represents a threshold. If the activity of every oscillator is below this threshold, then the global inhibitor will not receive any input. In this case $z \to 0$ and the oscillators will not receive any inhibition. If, on the other hand, the activity of at least one oscillator is above the threshold $\theta_{zx}$ then, the global inhibitor will receive input. In this case $z \to 1$, and each oscillator feels inhibition when $z$ is above the threshold $\theta_{zx}$. The parameter $\phi$ determines the rate at which the inhibitor reacts to such stimulation.

In summary, once an oscillator is active, it triggers the global inhibitor. This then inhibits the entire network as described in Eq. 1. On the other hand, an active oscillator spreads its activation to its nearest neighbors, again through (1), and from them to its further neighbors. In the next section, we give a number of properties of this system.

Besides boundaries, the oscillators on the grid are basically symmetrical. Boundary conditions may cause certain distortions to the stability of synchrous oscillations. Recently, Wang (1993) proposed a mechanism called *dynamic normalization* to ensure that each oscillator, whether it is in the interior or on a boundary, has equal overall connection weights from its neighbors. The dynamic normalization mechanism is adopted in the present model to form effective connections. For binary images (each pixel being either 0 or 1), the outcome of dynamic normalization is that an effective connection is established between two oscillators if and only if they are neighbors and both of them are activated by external stimulation. The network defined above can readily be applied for segmentation of binary images. For gray-level images (each pixel being in a certain value range), the following slight modification suffices to make the network applicable. An effective connection is established between two oscillators if and only if they are neighbors and the difference of their corresponding pixel values is below a certain threshold.

# 3 ANALYTICAL RESULTS

We have formally analyzed the network. Due to space limitations, we can only list the major conclusions without proofs. The interested reader can find the details in Terman and Wang (1994). Let us refer to a *pattern* as a connected region, and a *block* be a subset of oscillators stimulated by a given pattern. The following results are about singular solutions in the sense that we formally set $\varepsilon = 0$. However, as shown in (Terman and Wang, 1994), the results extend to the case $\varepsilon > 0$ sufficiently small.

**Theorem 1.** (*Synchronization*). The parameters of the system can be chosen so that all of the oscillators in a block always jump up simultaneously (synchronize). Moreover, the rate of synchronization is exponential.

**Theorem 2.** (*Multiple Patterns*) The parameters of the system and a constant $T$ can be chosen to satisfy the following. If at the beginning all the oscillators of the same block synchronize with each other and the temporal distance between any two oscillators belonging to two different blocks is greater than $T$, then (1) Synchronization within each block is maintained; (2) The blocks activate with a fixed ordering; (3) At most one block is in its active phase at any time.

**Theorem 3.** (*Desynchronization*) If at the beginning all the oscillators of the system lie not too far away from each other, then the condition of Theorem 2 will be satisfied after some time. Moreover, the time it takes to satisfy the condition is no greater than $N$ cycles, where $N$ is the number of patterns.

The above results are true with arbitrary number of oscillators. In summary, the network exhibits a mechanism, referred to as *selective gating*, which can be intuitively interpreted as follows. An oscillator jumping to its active phase opens a gate to quickly recruit the oscillators of the same block due to local connections. At the same time, it closes the gate to the oscillators of different blocks. Moreover, segmentation of different patterns is achieved very rapidly in terms of oscillation cycles.

# 4 COMPUTER SIMULATION

To illustrate how this network is used for scene segmentation, we have simulated a 20x20 oscillator network as defined by (1)-(4). We arbitrarily selected four objects (patterns): two **O**'s, one **H**, and one **I**; and they form the word **OHIO**. These patterns were simultaneously presented to the system as shown in Figure 3A. Each pattern is a connected region, but no two patterns are connected to each other.

All the oscillators stimulated (covered) by the objects received an external input $I = 0.2$, while the others have $I = -0.02$. The amplitude $\rho$ of the Gaussian noise is set to 0.02. Thus, compared to the external input, a 10% noise is included in every oscillator. Dynamic normalization results in that only two neighboring oscillators stimulated by a single pattern have an effective connection. The differential equations were solved numerically with the following parameter values: $\varepsilon = 0.02$, $\phi = 3.0$; $\gamma = 6.0$, $\beta = 0.1$, $K = 50$, $\theta_x = -0.5$, and $\theta_{zx} = \theta_{xz} = 0.1$. The total effective connections were normalized to 6.0. The results described below were robust to considerable changes in the parameters. The phases of all the oscillators on the grid were randomly initialized.

Fig. 3B-3F shows the instantaneous activity (snapshot) of the network at various stages of dynamic evolution. The diameter of each black circle represents the normalized $x$ activity of the corresponding oscillator. Fig. 3B shows a snapshot of the network a few steps after the beginning of the simulation. In Fig. 3B, the activities of the oscillators were largely random. Fig. 3C shows a snapshot after the system had evolved for a short time period. One can clearly see the effect of grouping and segmentation: all the oscillators belonging to the left **O** were entrained and had large activities. At the same time, the oscillators stimulated by the other three patterns had very small activities. Thus the left **O** was segmented from the rest of the input. A short time later, as shown in Fig. 3D, the oscillators stimulated by the right **O** reached high values and were separated from the rest of the input. Fig. 3E shows another snapshot after Fig. 3D. At this time, pattern **I** had its turn to be activated and separated from the rest of the input. Finally in Fig. 3F, the oscillators representing **H** were active and the rest of the input remained silent. This successive "pop-out" of the objects continued in a stable periodic fashion. To provide a complete picture of dynamic evolution, Fig. 3G shows the temporal evolution of each oscillator. Since the oscillators receiving no external input were inactive during the entire simulation process, they were excluded from the display in Fig. 3G. The activities of the oscillators stimulated by each object are combined together in the figure. Thus, if they are synchronized, they appear like a single oscillator. In Fig. 3G, the four upper traces represent the activities of the four oscillator blocks, and the bottom trace represents the activity of the global inhibitor. The synchronized oscillations within each object are clearly shown within just three cycles of dynamic evolution.

## 5   DISCUSSION

Besides neural plausibility, oscillatory correlation has a unique feature as an computational approach to the engineering of scene segmentation and figure/ground segregation. Due to the nature of oscillations, no single object can dominate and suppress the perception of the rest of the image permanently. The current dominant object has to give way to other objects being suppressed, and let them have a chance to be spotted. Although at most one object can dominant at any time instant, due to rapid oscillations, a number of objects can be activated over a short time period. This intrinsic dynamic process provides a natural and reliable representation of multiple segmented patterns.

The basic principles of selective gating are established for the network with lateral connections beyond nearest neighbors. Indeed, in terms of synchronization, more distant connections even help expedite phase entrainment. In this sense, synchronization with all-to-all connections is an extreme case of our system. With nearest-neighbor connectivity (Fig. 2), any isolated part of an image is considered as a segment. In an noisy image with many tiny regions, segmentation would result in too many small fragments. More distant connections would also provide a solution to this problem. Lateral connections typically take on the form of Gaussian distribution, with the connection strength between two oscillators falling off exponentially. Since global inhibition is superimposed to local excitation, two oscillators positively coupled may be desynchronized if global inhibition is strong enough. Thus, it is unlikely that all objects in an image form a single segment as the result of extended connections.

Due to its critical importance for computer vision, scene segmentation has been studied quite extensively. Many techniques have been proposed in the past (Haralick and Shapiro, 1985; Sarkar and Boyer, 1993). Despite these techniques, as pointed out by Haralick and Shapiro (1985), there is no underlying theory of image segmentation, and the techniques tend to be adhoc and emphasize some aspects while ignoring others. Compared to the traditional techniques for scene segmentation, the oscillatory correlation approach offers many unique advantages. The dynamical process is inherently parallel. While conventional computer vision algorithms are based on descriptive criteria and many adhoc heuristics, the network as exemplified in this paper performs computations based on only connections and oscillatory dynamics. The organizational simplicity renders the network particularly feasible for VLSI implementation. Also, continuous-time dynamics allows real time processing, desired by many engineering applications.

## Acknowledgments

DLW is supported in part by the NSF grant IRI-9211419 and the ONR grant N00014-93-1-0335. DT is supported in part by the NSF grant DMS-9203299LE.

## References

R. Eckhorn, et al., "Coherent oscillations: A mechanism of feature linking in the visual cortex?" *Biol. Cybern.*, vol. 60, pp. 121-130, 1988.

C.M. Gray, P. König, A.K. Engel, and W. Singer, "Oscillatory responses in cat visual cortex exhibit inter-columnar synchronization which reflects global stimulus properties," *Nature*, vol. 338, pp. 334-337, 1989.

R.M. Haralick and L.G. Shapiro, "Image segmentation techniques," Comput. Graphics Image Process., vol. 29, pp. 100-132, 1985.

P. König and T.B. Schillen, "Stimulus-dependent assembly formation of oscillatory responses: I. Synchronization," *Neural Comput.*, vol. 3, pp. 155-166, 1991.

S. Sarkar and K.L. Boyer, "Perceptual organization in computer vision: a review and a proposal for a classificatory structure," IEEE Trans. Syst. Man Cybern., vol. 23, 382-399, 1993.

D. Somers, and N. Kopell, "Rapid synchronization through fast threshold modulation," *Biol. Cybern,* vol. 68, pp. 393-407, 1993.

O. Sporns, G. Tononi, and G.M. Edelman, "Modeling perceptual grouping and figure-ground segregation by means of active reentrant connections," *Proc. Natl. Acad. Sci. USA*, vol. 88, pp. 129-133, 1991.

D. Terman and D.L. Wang, "Global competition and local cooperation in a network of neural oscillators," *Physica D*, in press, 1994.

C. von der Malsburg, "The correlation theory of brain functions," Internal Report 81-2, Max-Planck-Institut for Biophysical Chemistry, Göttingen, FRG, 1981.

C. von der Malsburg and W. Schneider, "A neural cocktail-party processor," *Biol. Cybern.*, vol. 54, pp. 29-40, 1986.

D.L. Wang, "Modeling global synchrony in the visual cortex by locally coupled neural oscillators," *Proc. 15th Ann. Conf. Cognit. Sci. Soc.*, pp. 1058-1063, 1993.

D.L. Wang, "Emergent synchrony in locally coupled neural oscillators," *IEEE Trans. on Neural Networks*, in press, 1995.

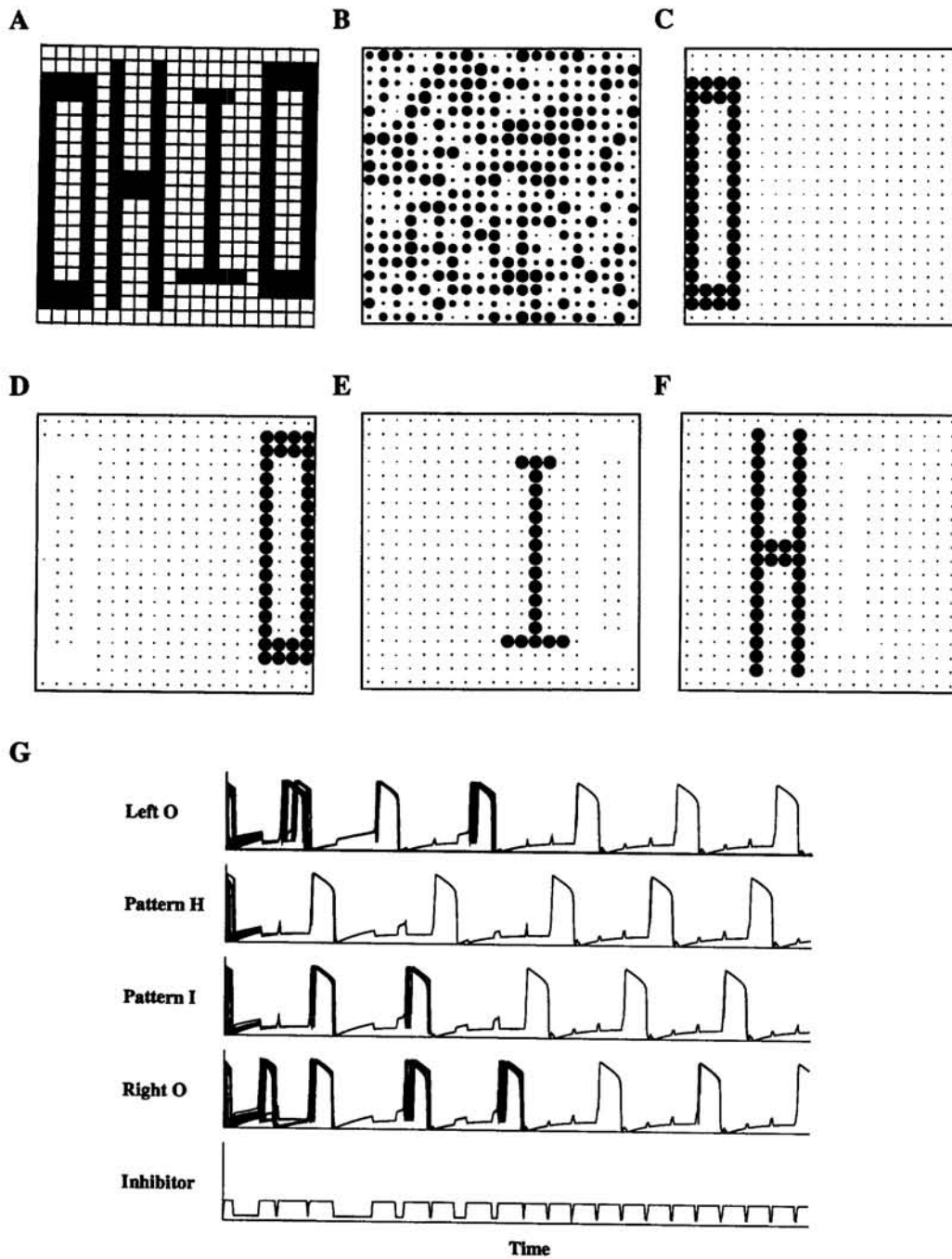

Figure 3. **A** An image composed of four patterns which were presented (mapped) to a 20x20 grid of oscillators. **B** A snapshot of the activities of the oscillator grid at the beginning of dynamic evolution. **C** A snapshot taken shortly after the beginning. **D** Shortly after C. **E** Shortly after D. **F** Shortly after E. **G** The upper four traces show the combined temporal activities of the oscillator blocks representing the four patterns, respectively, and the bottom trace shows the temporal activity of the global inhibitor. The simulation took 8,000 integration steps.